# APPLICATIONS OF ERROR BACK-PROPAGATION TO PHONETIC CLASSIFICATION

Hong C. Leung & Victor W. Zue
Spoken Language Systems Group
Laboratory for Computer Science
Massachusetts Institute of Technology
Cambridge, MA 02139

## ABSTRACT

This paper is concerced with the use of error back-propagation in phonetic classification. Our objective is to investigate the basic characteristics of back-propagation, and study how the framework of multi-layer perceptrons can be exploited in phonetic recognition. We explore issues such as integration of heterogeneous sources of information, conditions that can affect performance of phonetic classification, internal representations, comparisons with traditional pattern classification techniques, comparisons of different error metrics, and initialization of the network. Our investigation is performed within a set of experiments that attempts to recognize the 16 vowels in American English independent of speaker. Our results are comparable to human performance.

Early approaches in phonetic recognition fall into two major extremes: heuristic and algorithmic. Both approaches have their own merits and shortcomings. The heuristic approach has the intuitive appeal that it focuses on the linguistic information in the speech signal and exploits acoustic-phonetic knowledge. However, the weak control strategy used for utilizing our knowledge has been grossly inadequate. At the other extreme, the algorithmic approach relies primarily on the powerful control strategy offered by well-formulated pattern recognition techniques. However, relatively little is known about how our speech knowledge accumulated over the past few decades can be incorporated into the well-formulated algorithms. We feel that artificial neural networks (ANN) have some characteristics that can potentially enable them to bridge the gap between these two extremes. On the one hand, our speech knowledge can provide guidance to the structure and design of the network. On the other hand, the self-organizing mechanism of ANN can provide a control strategy for utilizing our knowledge.

In this paper, we extend our earlier work on the use of artificial neural networks for phonetic recognition [2]. Specifically, we focus our investigation on the following sets of issues. First, we describe the use of the network to integrate heterogeneous sources of information. We will see how classification performance improves as more

information is available. Second, we discuss several important factors that can substantially affect the performance of phonetic classification. Third, we examine the internal representation of the network. Fourth, we compare the network with two traditional classification techniques: K-nearest neighbor and Gaussian classification. Finally, we discuss our specific implementations of back-propagation that yield improved performance and more efficient learning time.

# EXPERIMENTS

Our investigation is performed within the context of a set of experiments that attempts to recognize the 16 vowels in American English independent of speaker. The vowels are excised from continuous speech and they can be preceded and followed by any phonemes, thus providing a rich environment to study contextual influence. We assume that the locations of the vowels have been detected. Given a time region, the network determines which one of the 16 vowels was spoken.

## CORPUS

As Table 1 shows, our training set consists of 20,000 vowel tokens, excised from 2,500 continuous sentences spoken by 500 male and female speakers. The test set consists of about 2,000 vowel tokens, excised from 250 sentences spoken by 50 different speakers. All the data are extracted from the TIMIT database, which has a wide range of American dialectical variations [1]. The speech signal is represented by spectral vectors obtained from an auditory model [4]. Speaker and energy normalization are also performed [5].

|  | Tokens | Sentences | Speakers (M/F) |
|---|---|---|---|
| Training | 20,000 | 2500 | 500 (350/150) |
| Testing | 2,000 | 250 | 50 (33/17) |

Table 1: Corpus extracted from the TIMIT database.

## NETWORK STRUCTURE

The structure of the network we have examined most extensively has 1 hidden layer as shown in Figure 1. It has 16 output units, with one unit for each of the 16 vowels. In order to capture dynamic information, the vowel region is divided into three equal subregions. An average spectrum is then computed in each subregion. These 3 average spectra are then applied to the first 3 sets of input units. Additional sources of information, such as duration and local phonetic contexts, can also be made available to the network. While spectral and durational inputs are continuous and numerical, the contextual inputs are discrete and symbolic.

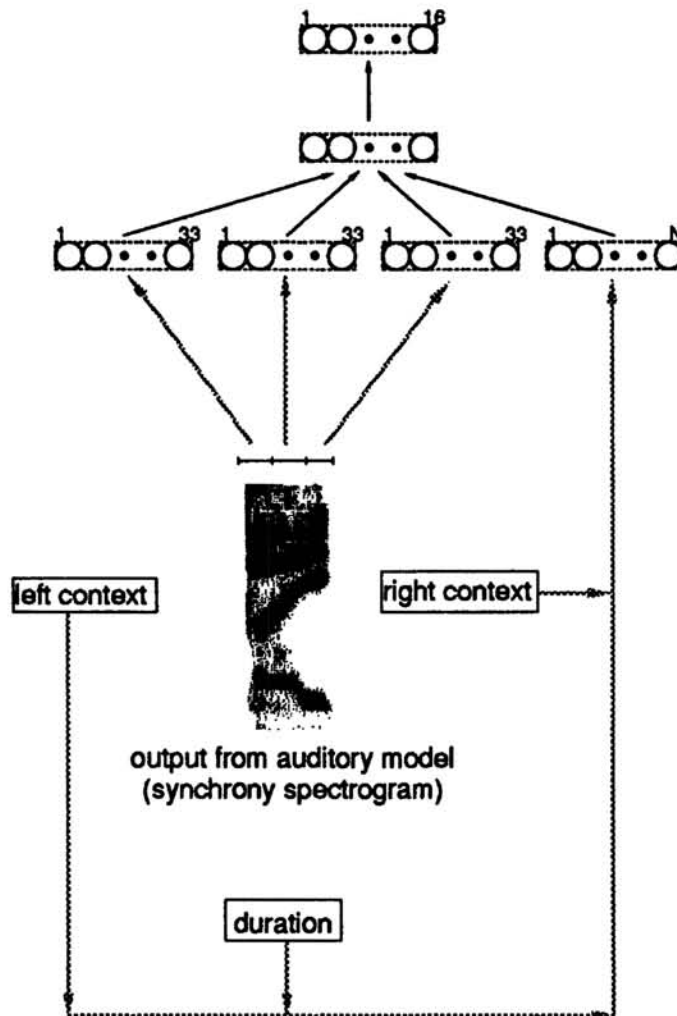

Figure 1: Basic structure of the network.

## HETEROGENEOUS INFORMATION INTEGRATION

In our earlier study, we have examined the integration of the Synchrony Envelopes and the phonetic contexts [2]. The Synchrony Envelopes, an output of the auditory model, have been shown to enhance the formant information. In this study, we add additional sources of information. Figure 2 shows the performance as heterogeneous sources of information are made available to the network. The performance is about 60% when only the Synchrony Envelopes are available. The performance improves to 64% when the Mean Rate Response, a different output of the auditory model which has been shown to enhance the temporal aspects of the speech signal, is also available. We can also see that the performance improves consistently to 77% as durational and contextual inputs are provided to the network. This experiment suggests that the network is able to make use of heterogeneous sources of information, which can be numerical and/or symbolic.

One may ask how well human listeners can recognize the vowels. Experiments have been performed to study how well human listeners agree with each other when they can only listen to sequences of 3 phonemes, i.e. the phoneme before the vowel, the vowel itself, and the phoneme after the vowel [3]. Results indicate that the average agreement among the listeners on the identities of the vowels is between 65% and 70%.

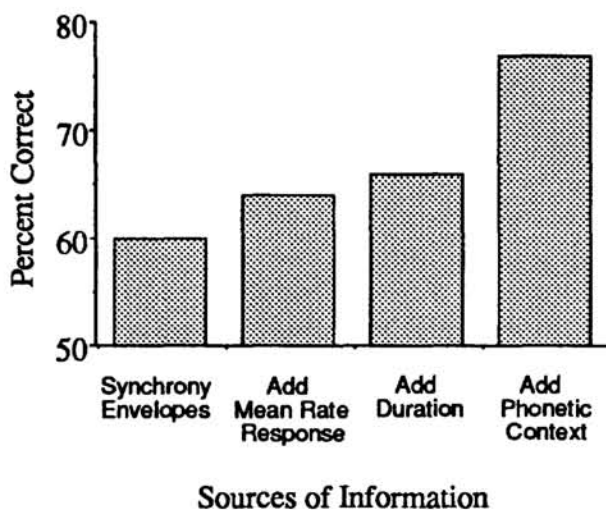

Figure 2: Integration of heterogeneous sources of information.

## PERFORMANCE RESULTS

We have seen that one of the important factors for the network performance is the amount of information available to the network. To gain additional insights about how the network performs under different conditions, several experiments were conducted using different databases. In these and the subsequent experiments we describe in this paper, only the Synchrony Envelopes are available to the network.

Table 2 shows the performance results for several recognition tasks. In each of these tasks, the network is trained and tested with independent sets of speech data. The first task recognizes vowels spoken by one speaker and excised from the /b/-vowel-/t/ environment, spoken in isolation. This recognition task is relatively straightforward, resulting in perfect performance. In the second experiment, vowel tokens are extracted from the same phonetic context, but spoken by 17 male and female speakers. Due to inter-speaker variability, the accuracy degrades to 86%. The third task recognizes vowels spoken by one speaker and excised from an unrestricted context, spoken continuously. We can see that the accuracy decreases further to 70%. Finally, data from the TIMIT database are used, spoken by multiple speakers. The accuracy drops to 60%. These results indicate that a substantial difference in performance can be expected under different conditions, depending on whether the task is speaker-independent, what is the restriction on the phonetic

| Speakers(M/F) | Context | Training Tokens | Percent Correct | Remark |
|---|---|---|---|---|
| 1(1/0) | b __ t | 64 | 100 | isolated |
| 17(8/9) | b __ t | 256 | 86 | isolated |
| 1(1/0) | * __ * | 3,000 | 70 | continuous |
| 500(350/150) | * __ * | 20,000 | 60 | continuous |

Table 2: Performance for different tasks, using only the synchrony spectral information. "*" stands for any phonetic contexts.

contexts, whether the speech material is spoken continuously, and how much data are used to train the network.

## INTERNAL REPRESENTATION

To understand how the network makes use of the input information, we examined the connection weights of the network. A vector is formed by extracting the connections from all the hidden units to one output unit as shown in Figure 3a. The same process is repeated for all output units to obtain a total of 16 vectors. The correlations among these vectors are then examined by measuring the inner products or the angles between them. Figure 3b shows the distribution of the angles after the network is trained, as a function of the number of hidden units. The circles represent the mean of the distribution and the vertical bars stand for one standard deviation away from the mean. As the number of hidden units increases, the distribution becomes more and more concentrated and the vectors become increasingly orthogonal to each other.

The correlations of the connection weights before training were also examined, as shown in Figure 3c. Comparing parts (b) and (c) of Figure 3, we can see that the distributions before and after training overlap more and more as the number of hidden units increases. With 128 hidden units, the two distributions are actually quite similar. This leads us to suspect that perhaps the connection weights between the hidden and the output layer need not be trained if we have a sufficient number of hidden units.

Figure 4a shows the performance of recognizing the 16 vowels using three different techniques: (i) train all the connections in the network, (ii) fix the connections between the hidden and output layers after random initialization and train only the connections between the input and hidden layers, and (iii) fix the connections between the input and hidden layers and train only the connections between the hidden and output layers. We can see that with enough hidden units, training only the connections between the input and the hidden layers achieves almost the same performance as training all the connections in the network. We can also see that

for the same number of hidden units, training only the connections between the input and the hidden layer can achieve higher performance than training only the connections between the hidden and the output layer.

Figure 4b compares the three training techniques for 8 vowels, resulting in 8 output units only. We can see similar characteristics in both parts (a) and (b) of Figure 4.

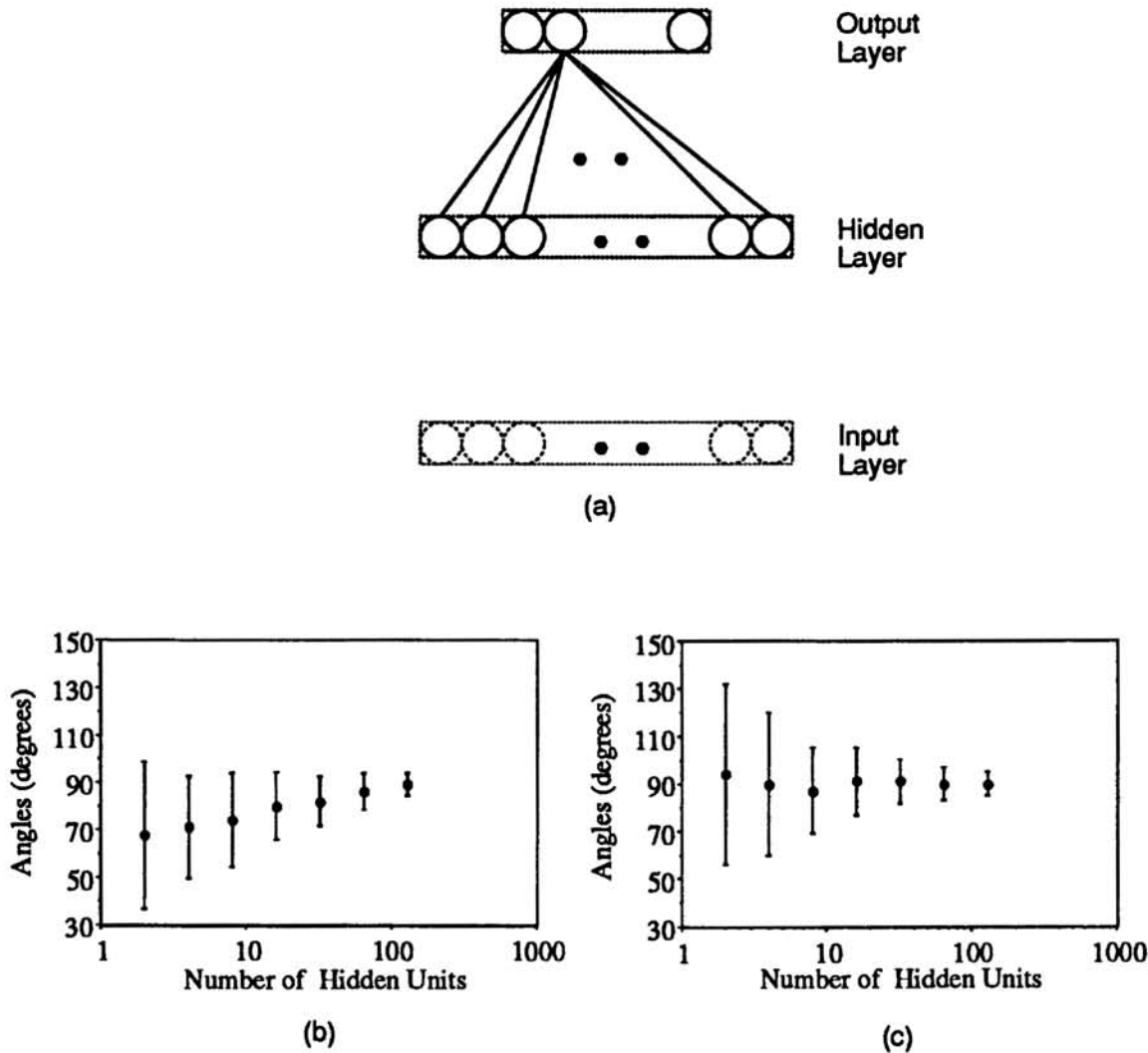

Figure 3: (a) Correlations of the vectors from the hidden to output layers are examined. (b) Distribution of the angles between these vectors after training. (c) Distribution of the angles between these vectors before training.

## COMPARISONS WITH TRADITIONAL TECHNIQUES

One of the appealing characteristics of back-propagation is that it does not assume any probability distributions or distance metrics. To gain further insights, we compare with two traditional pattern classification techniques: K-nearest neighbor (KNN) and multi-dimensional Gaussian classifiers.

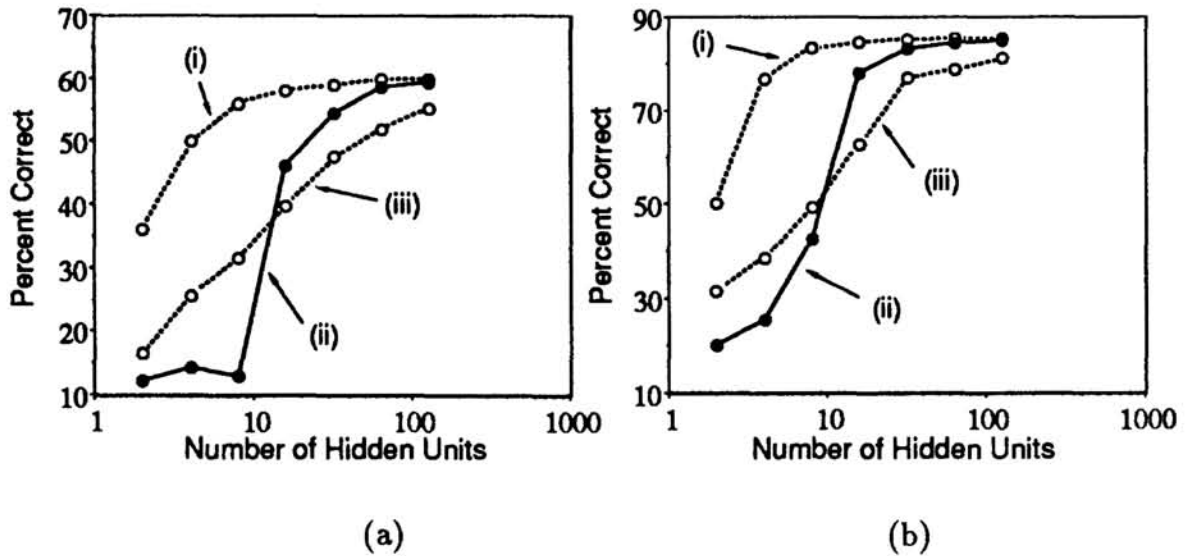

Figure 4: Performance of recognizing (a) 16 vowels, (b) 8 vowels when (i) all the connections in the network are trained, (ii) only the connections between the input and hidden layers are trained, and (iii) only the connections between the hidden and output layers are trained.

Figure 5a compares the performance results of the network with those of KNN, for different amounts of training tokens. Again, only the Synchrony Envelopes are made available to the network, resulting in input vectors of 100 dimensions. Each cluster of crosses corresponds to performance results of ten networks, each one randomly initialized differently. Due to different initialization, a fluctuation of 2% to 3% is observed even for the same training size. For comparison, we perform KNN using the Euclidean distance metric. For each training size, we run KNN 6 times, each one with a different K, which is chosen to be proportional to the square root of the number of training tokens, N. For simplicity, Figure 5a shows results for only 3 different values of K: $(i)$ $K = \sqrt{N}$, $(ii)$ $K = 10\sqrt{N}$, $and$ $(iii)$ $K = 1$. In this experiment, we have found that the performance is the best when $K = \sqrt{N}$ and is the worst when $K = 1$. We have also found that up to 20,000 training tokens, the network consistently compares favorably to KNN. It is possible that the network is able to find its own distance metric to achieve better performance.

Since the true underlying probability distribution is unknown, we assume multi-dimensional Gaussian distribution in the second experiment. (i) We use the full covariance matrix, which has $100x100$ elements. To avoid problems with singularity, we obtain results only for large number of training tokens. (ii) We use the diagonal covariance matrix which has non-zero elements only along the diagonal. We can see from Figure 5b that the network compares favorably to the Gaussian classifiers. Our results also suggest that the Gaussian assumption is invalid.

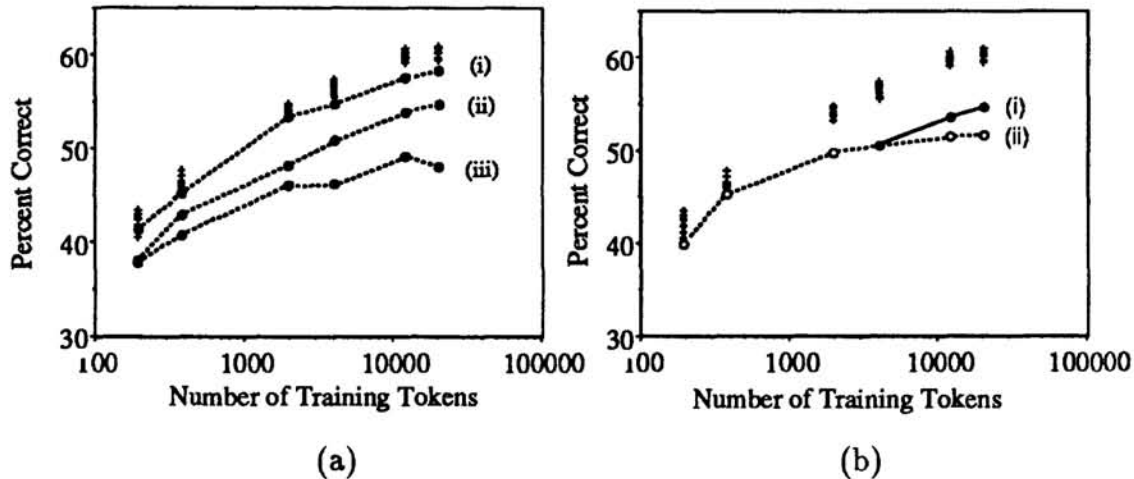

Figure 5: (a) Comparison with KNN for different values of K (See text). (b) Comparison with Gaussian classification when using the (i) full covariance matrix, and (ii) diagonal covariance matrix. Each cluster of 10 crosses corresponds to the results of 10 different networks, each one randomly initialized.

## ERROR METRIC AND INITIALIZATION

In order to take into account the classification performance of the network more explicitly, we have introduced a weighted mean square error metric [2]. By modulating the mean square error with weighting factors that depend on the classification performance, we have shown that the rank order statistics can be improved. Like simulated annealing, gradient descent takes relatively big steps when the performance is poor, and takes smaller and smaller steps as the performance of the network improves.

Results also indicate that it is more likely for a unit output to be initially in the saturation regions of the sigmoid function if the network is randomly initialized. This is not desirable since learning is slow when a unit output is in a saturation region. Let the sigmoid function goes from −1 to 1. If the connection weights between the input and the hidden layers are initialized with zero weights, then all the hidden unit outputs in the network will initially be zero, which in turn results in zero output values for all the output units. In other words, all the units will initially operate at the center of the transition region of the sigmoid function, where learning is the fastest. We call this method center initialization (CI).

Parts (a) and (b) of Figure 6 compare the learning speed and performance, respectively, of the 3 different techniques: (i) mean square error (MSE), (ii) weighted mean square error (WMSE), and (iii) center initialization (CI) with WMSE. We can see that both WMSE and CI seem to be effective in improving the learning time and the performance of the network.

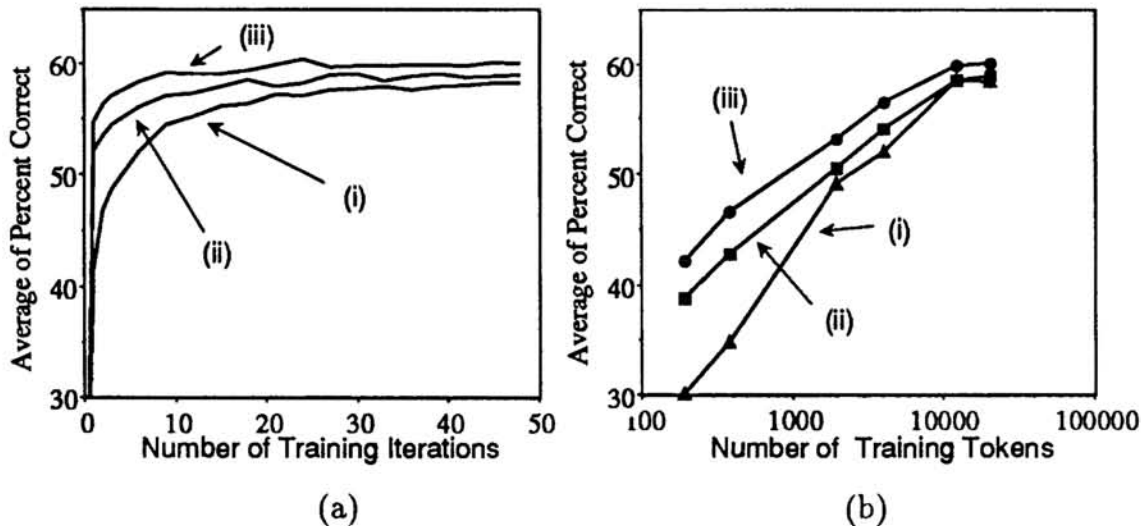

Figure 6: Comparisons of the (a) learning characteristics and, (b) performance results, for the 3 different techniques: (i) MSE, (ii) WMSE, and (iii) CI with WMSE. Each point corresponds to the average of 10 different networks, each one initialized randomly.

## SUMMARY

In summary, we have described a set of experiments that were designed to help us get a better understanding of the use of back-propagation in phonetic classification. Our results are encouraging and we are hopeful that artificial neural networks may provide an effective framework for utilizing our acoustic-phonetic knowledge in speech recognition.

# References

[1] Fisher, W.E., Doddington, G.R., and Goudie-Marshall, K.M., "The DARPA Speech Recognition Research Database: Specifications and Status," *Proceedings of the DARPA Speech Recognition Workshop* Report No. SAIC-86/1546, February, 1986.

[2] Leung, H.C., "Some phonetic recognition experiments using artificial neural nets," ICASSP-88, 1988.

[3] Phillips, M.S., "Speaker independent classification of vowels and diphthongs in continuous speech," *Proc. of the 11th International Congress of Phonetic Sciences,* Estonia, USSR, 1987.

[4] Seneff S., "A computational model for the peripheral auditory system: application to speech recognition research," *Proc. ICASSP*, Tokyo, 1986.

[5] Seneff S., "Vowel recognition based on 'line-formants' derived from an auditory-based spectral representation," *Proc. of the 11th International Congress of Phonetic Sciences,* Estonia, USSR, 1987.
